# A Systematic Study of the Input/Output Properties of a 2 Compartment Model Neuron With Active Membranes

**Paul Rhodes**
University of California, San Diego

## ABSTRACT

The input/output properties of a 2 compartment model neuron are systematically explored. Taken from the work of MacGregor (MacGregor, 1987), the model neuron compartments contain several active conductances, including a potassium conductance in the dendritic compartment driven by the accumulation of intradendritic calcium. Dynamics of the conductances and potentials are governed by a set of coupled first order differential equations which are integrated numerically. There are a set of 17 internal parameters to this model, specificying conductance rate constants, time constants, thresholds, etc.

To study parameter sensitivity, a set of trials were run in which the input driving the neuron is kept fixed while each internal parameter is varied with all others left fixed.

To study the input/output relation, the input to the dendrite (a square wave) was varied (in frequency and magnitude) while all internal parameters of the system were left fixed, and the resulting output firing rate and bursting rate was counted.

The input/output relation of the model neuron studied turns out to be much more sensitive to modulation of certain dendritic potassium current parameters than to plasticity of synapse efficacy per se (the amount of current influx due to synapse activation). This would in turn suggest, as has been recently observed experimentally, that the potassium current may be as or more important a focus of neural plasticity than synaptic efficacy.

## INTRODUCTION

In order to model biologically realistic neural systems, we will ultimately be seeking to construct networks with thousands of neurons and millions of interconnections. It is therefor desireable to employ basic units with sufficient computational simplicity to make meaningful simulations tractable, yet with sufficient fidelity to biological neurons that we may retain a hope of gleaning by these simulations something about the activity going on during biological information processing.

The types of neuron models employed in the computational neuroscience literature range from binary threshold units to sigmoid transfer functions to 1500 compartment neurons with Hodgkin-Huxley kinetics for a whole set of active conductances and spines with rich internal structure. In principle, a model neuron's functional participation in the operation of a network may be fully characterized by a complete description of its transfer function, or input-output relation. This relation would necessarily be parameterized by a host of internal variables (which would include conductance rate constants and parameters defining the neuron's morphology) as well as a very rich space characterizing possible variations in input (including location of input in dentritic tree). In learning to judge which structural elements of highly realistic models must be preserved and which may be simplified, one approach will be to test the degree to which the input-output relation of the simplified neuron (given a physiologically relevant parameter range and input space) is sufficiently close to the input-output properties of the highly realistic model.

To define 'sufficiently close', we will ultimately refer to the operation of the network as a whole as follows: the transfer function of a simplified neuron model will be considered 'sufficiently close' to a more realistic neuron model if a chosen information processing task carried out by the overall network is performed by a network built up of the simplified neurons in a manner close to that observed in a network of the more realistic neurons.

We propose to begin by exploring the input/output properties of a greatly simplified 2 compartment model neuron with active conductances. Even in this very simple structure there are many (17) internal parameters for things like time constants and activation rates of currents. We wish to understand the parameter sensitivity of this model system and characterize its input-output relation.

## 1.0  DESCRIPTION OF THE MODEL NEURON

THE MODEL NEURON CONSISTS OF A SOMA WITH A VOLTAGE-GATED POTASSIUM CONDUCTANCE AND A SINGLE COMPARTMENT DENDRITE WITH A VOLTAGE-GATED CALCIUM CONDUCTANCE AND A [CA]-GATED POTASSIUM CONDUCTANCE

We will choose for this study a simple model neuron described by MacGregor (1987). It possesses a single compartment dendrite. This is viewed as a crude approximation to the lumped reduction of a dendritic tree. In this approximation, we are neglecting spatial and temporal summing of individual synaptic EPSP's distributed over a dendritic tree, as well as the spatial and temporal dispersion (smearing) due to transmission to the soma. The individual inputs we will be using are large enough to drive the soma to firing, and so would represent the summation of many relatively simultaneous individual EPSP's, perhaps as from the set of contacts upon a neuron's dendritic tree made by the arborization of one different axon. The dendritic membrane possesses a potassium conductance gated by intradendritic calcium concentration and a voltage gated calcium conductance. The soma contains its own voltage-gated potassium channels and membrane time constants. Electrical connection between soma and dendrite is expressed by an input impedance in each direction. The soma fires an action potential, simply expressed by raising its voltage to 50 mv for one msec after its internal voltage has been

driven to firing threshold. Calcium accumulation in the dendrite is modelled assuming accumulation proportional to calcium conductance. Calcium conductance itself increases in proportion to the difference between the dendrite's voltage and a threshold, and calcium is removed from the dendrite by means of an exponential decay. This system is modelled by a set of coupled first order differential equations as follows:

## 1.1 THE SET OF EQUATIONS GOVERNING THE DYNAMIC VARIABLES OF THIS MODEL

The soma's voltage ES is governed by:

$$dES/dt=\{-ES+SOMAINPUT+GDS*(ED-ES)+GKS*(EK-ES)\}/TS$$

where SOMAINPUT is obtained by dividing the input current by the total resting conductance of the dendrite (therefor it has units of voltage). GDS is proportional to input resistance from dendrite to soma, and multiplies the difference between the dendrite's voltage ED and the soma's voltage ES; GKS is the soma's aggregate potassium conductance (modelled below); EK is the voltage of the potassium battery (assumed constant at -10mv); and TS is the soma's time constant. All potentials are relative to resting potential, and all conductances are dimensionless.

The dendrite's voltage ED is govened by:

$$dED/dt=\{-ED+DENDINPUT+GSD*(ES-ED)+GCA*(ECA-ED)+ GKD*(EK-ED)\}/TD$$

where DENDINPUT is obtained by dividing the input current by the total resting conductance of the dendrite and so has units of voltage. GSD is proportional to the input resistance from soma to dendrite, and hence multiplies the difference between ES and ED; GCA is the dendrite's calcium conductance (modelled below), ECA is the calcium battery (assumed constant at 50mv), and GKD is proportional to the dendrite's potassium conductance (modelled below). All potentials are relative to resting potential.

The soma's voltage is raised artificially to 50mv for 1 msec after the soma's voltage exceeds a (fixed) threshold, thus simplifying the action potential.

The potassium conductance in the soma, GKS, is governed by :

$$dGKS/dt=\{-GKS+S*B\}/TGK$$

where S is 1 if an action potential has just fired and 0 otherwise, B is an activation rate constant governing the rate of increase of potassium conductance, and TGK is the time constant of the potassium conductance decay. This rather simplified picture of potassium conductance will be replaced by a more realistic version with a Markov state model of the potassium channel in a subsequent publication in preparation. For the present investigation then we are modelling the voltage dependence of the potassium conductance by the following: potassium conductance builds up by a fixed amount (proportional to B/TGK) during each action potential, and thereafter decays exponentially with time constant TGK.

The dendrite's calcium conductance is governed by:

dGCA/dt={-GCA+D*(ED-CSPIKETHRESH)}/TGCA      ED>CSPIKETHRESH
dGCA/dt={-GCA/TGCA}                          ED<CSPIKETHRESH

where CSPIKETHRESH is the minimum dendritic voltage above which calcium conducting channels begin to be opened, D is an activation rate governing the rate of increase in calcium conductance, and TGCA is the time constant assumed to govern conductance decay when voltage is below threshold.

The dendrite's internal calcium concentration [CA] is governed by:

d[CA]/dt={-[CA]+A*GCA}/TCA

where TCA is the time constant for the removal of internal CA, and A is a parameter governing the accumulation rate of increase of internal CA for a given conductance and time constant.  A is inversely proportional to the effective relevant volume in which calcium is accumulating.  An increase in internal calcium buffer would decrease the parameter A.

Finally, the dendrite's potassium conductance is governed by:

dGKD/dt={-GKD+BD}/TGKD                       [CA]>CALCTHRESH
dGKD/dt={-GKD}/TGKD                          [CA]<CALCTHRESH

where CALCTHRESH is the internal calcium concentration threshold above which the calcium gated potassium channel begins to open, BD is the parameter governing the rate of increase of dendritic potassium conductance, and TGKD is the time constant governing the exponential decay of potassium conductance.

**This entire system of equations is taken from the work of MacGregor (MacGregor, 1987).**

The system of coupled first order differential equations is integrated using the exponential method, also discussed in MacGregor.  Generally a 1 msec timestep is used, with a smaller timestep of .1 msec used for the relaxation between the dendritic voltage ED and the somatic voltage ES.

## 2.0  THE EFFECT OF CHANGES IN PARAMETERS (TIME CONSTANTS, CONDUCTANCE RATES, ETC.) ON THE MODEL NEURON'S INPUT-OUTPUT PROPERTIES WILL BE EXPLORED

As is clear from a review of the above set of interrelated equations governing the dynamics of the state variables of the model neuron, there are quite a few externally specified parameters (17) even in such a simple model.  Presumably the thresholds are fairly well measureable, and the rate constants and time constants may be specified by measurement of time courses in patch clamp experiments.  We are nevertheless dealing with parameters of which some are thought to be variable and which are probably

modulated explicitly by normal mechanisms in neurons. Therefor we wish to explore the effect that variation of any of these parameters has on the input-output properties of the model neuron. **In fact, we will find indication that the modulation of these parameters, in particular the rate constants governing the dendritic potassium current and internal calcium accumulation, may be very effective targets of neural plasticity. We find that the neuron's input-output properties are more sensitive to these parameters than to modulation of the efficacy of the synapse strength per se.**

## 2.1 PROTOCOL FOR SYSTEMATIC EXPLORATION OF THE EFFECT OF VARIATION IN THE MODEL'S PARAMETERS ON THE INPUT-OUTPUT PROPERTIES OF THE MODEL NEURON

We started with the parameters all set to a set of benchmarks and drove the neuron with a constant input to the dendrite. (We could have driven the soma instead, or both soma and dendrite, and we could have chosen more complex input streams. See below for trials where we systematically vary the input but the parameter values are held steady.) The input was a steady command input of 35mv. The values of all the benchmark parameters are given in Table 1.

We then systematically halved and doubled each of the 17 parameters in turn, while leaving all other parameters fixed. Note that in all cases and in fact with any driving input this model neuron fires in bursts. This is due to the long time course of the potassium current in the dendrite, which enforces a long refractory period (about 40-80msec) even during continuous stimulation.

## 2.2 RESULTS OF SYSTEMATIC VARIATION OF PARAMETERS OF MODEL NEURON

**The results are summarized in the notes to Table 1.** Following are several observations about the different parameters' varying degree of efficacy in modulation of the input-output function.

1) The most striking finding is that variation of the activation rate of the potassium current, particularly the potassium current in the dendrite, is the most effective means of modulating the input-output properties of the model neuron. The transfer function is 250% more sensitive to an increase in the [CA]-gated dendritic potassium current activation rate than it is to an increase in synaptic efficacy per se.

2) Changing the **time constant** of the [CA]-gated potassium current in the dendrite is the only parameter change which effectively modulates the number of **bursts per second** (see Figure 1). Changing the time constant of the voltage-gated potassium current in the soma, does not have any effect on the number of bursts per second.

## 3.0   MEASUREMENT OF THE INPUT/OUTPUT RELATION OF THE MODEL NEURON

The input/output relation was determined by the following protocol: The input was supplied in the form of a square wave of current injected into the dendritic compartment, and the frequency of the pulses and their magnitude was systematically varied.

The output of the soma, in the form of action potentials fired per second, was plotted against the input rate, defined as the product of the square wave frequency and the magnitude of the injected current. The duration of pulses was kept fixed at 20 msec (but see below), all internal parameters were fixed at their benchmark levels.

### 3.1   THE SHAPE OF THE INPUT/OUTPUT RELATION

Figure 2 depicts the above described plot in the case where all the internal parameters were fixed at purported "benchmark" values except for the parameters governing intradendritic calcium accumulation.. It is clearly not strictly monotonic (there are resonance points) though a smoothed version is monotonic, and it does not faithfully render a sigmoid.

### 3.2   THE INPUT/OUTPUT RELATION IS UNCHANGED IF THE SQUARE SHAPE OF THE EPSP DRIVING THE DENDRITE IS REPLACED BY AN ALPHA FUNCTION

The trials in this study were largely conducted using a square wave as the input driving the dendritic compartment. In order to check whether the unphysical square shape of the envelope of this current injection was coloring the results, the input/output relation was measured in a set of trials wherein the alpha function commonly used to model the time course of EPSP's replaced the square pulse. The total current injected per pulse was kept uniform. **The results, shown in Figure 3, are surprising: The input/output relation was almost completely unaltered by the substitution. This suggests that the detailed shape and fourier spectrum of the time course of synaptic input has nearly no effect of the neuron's output.** Thus it is suggested that very adequate models can be built without the need for a strict modelling of the synaptic EPSP. I expect this effect is due to the temporal integration ongoing in the summation of input to this system, which blurs the exact shape of any input envelope.

### 3.3   MODULATION OF THE INPUT/OUTPUT RELATION BY VARIATION OF INTERNAL MODEL PARAMTERS

Figure 1 portrays the input/output relation measured in three cases in which all internal parameters are identical except the rate of accumulation of intradendric calcium. The lower curve is the case where the calcium accumulation rate is highest. Since [Ca] accumulation drives the dendritic potassium current, the activation of which in turn hyperpolarizes the dendrite and thus indirectly suppresses firing in the soma, we expect output in this case to be lower for a given input as is indeed the result observed. Note that the parameter being varied would be expected to be inversely proportional to the amount of available intradendritic calcium buffer. **Hence the amount of**

**intradendritic buffer has a profound ability to modulate the transfer
function of the system.**

## 4.0  CONCLUSIONS

As regards the shape of the transfer function itself, we have found it to be non-
monotonic (there are resonance points) unless it is smoothed.  The shape of the transfer
function appears little effected by the envelope of the EPSP (i.e. square pulse input
produces nearly the same transfer function as the case where alpha functions are
substituted for the square pulses in  modelling the EPSP).

A parameter sensitivity analysis of a 2 compartment model neuron with active
membranes reveals some unexpected results.  For example, the input/output (transfer)
function of the neuron is 250% more sensitive to the activation rate of the [CA]-gated
dendritic potassium current than it is to synaptic efficacy per se.  This in turn suggests
that, as has indeed been observed (Alkon et al, 1988; Hawkins, 1989; Olds et al, 1989),
nature might employ mechanisms other than simply increasing synaptic conductance
during the EPSP to enhance the efficacy of the transfer function.

Alkon, D.L. et al, J. Neurochemistry, Volume 51, 903, (1988).

Hawkins, R. D. in Computational Models of Learning in Simple Neural Systems,
Hawkins and Bower, Eds., Academic Press, (1989).

MacGregor, R., Neural and Brain Modelling, Academic Press, (1988).

Olds, J. L. et al, Science, Volume 245, 866, (1989).

TABLE 1

RESULTS OF PARAMETER SENSITIVITY ANALYSIS

PROTOCOL:  EACH OF THE 17 INTERNAL PARAMETERS OF THE MODEL
NEURON WAS VARIED IN TURN, WHILE ALL THE OTHERS WERE KEPT
FIXED AT BENCHMARK VALUES.  THE DENDRITE WAS DRIVEN IN
EACH CASE WITH A STEADY FIXED INPUT AND THE RESULTING
BURSTING RATE AND FIRING RATE WAS COUNTED.  IN THE FINAL
TRIAL, ALL THE PARAMETERS WERE LEFT FIXED AND THE INPUT
MAGNITUDE WAS VARIED, TO SIMULATE FOR COMPARISON THE
EFFECT OF MODULATION OF SYNAPTIC EFFICACY.

| PARAMETER | SYMBOL | | VALUE | BURSTS SEC | SPIKES/ BURST | FIRING FREQ. | FIRING FREQ. AS % OF BENCHMARK |
|---|---|---|---|---|---|---|---|
| SOMATIC MEMBRANE TIME CONSTANT | TS | BENCHMARK | 5.0 | 13.51 | 2 | 27.03 | 100.0% |
| | | LOW | 2.5 | 13.70 | 2 | 27.40 | 101.4% |
| | | HIGH | 10.0 | 12.82 | 2 | 25.64 | 94.9% |
| DENDRITIC MEMBRANE TIME CONSTANT | TD | BENCHMARK | 5.0 | 13.51 | 2 | 27.03 | 100.0% |
| | | LOW | 2.5 | 13.51 | 2 | 27.03 | 100.0% |
| | | HIGH | 10.0 | 12.66 | 2 | 25.32 | 93.7% |

| PARAMETER | SYMBOL | | VALUE | BURSTS SEC | SPIKES/ BURST | FIRING FREQ. | FIRING FREQ. AS % OF BENCHMARK |
|---|---|---|---|---|---|---|---|
| THRESHOLD FOR [CA]-GATED POTASSIUM CURRENT IN DENDRITE (1) | CALCTHRESH | BENCHMARK LOW HIGH | 20.0 10.0 40.0 | 13.51 12.82 13.51 | 2 1 3 | 27.03 12.82 40.54 | 100.0% 47.4% 150.0% |
| ACTIVATION RATE OF SOMATIC POTASSIUM CURRENT (2) | B | BENCHMARK LOW HIGH | 33.0 16.5 66.0 | 13.51 12.99 13.51 | 2 3 1 | 27.03 38.96 13.51 | 100.0% 144.2% 50.0% |
| ACTIVATION RATE OF DENDRITIC [CA]-GATED POTASSIUM CURRENT | BD | BENCHMARK LOW HIGH | 75.0 37.5 150.0 | 13.51 12.35 13.16 | 2 4 2 | 27.03 49.38 26.32 | 100.0% 182.7% 97.4% |
| TIME CONSTANT OF SOMATIC POTASSIUM CURRENT (2) | TGK | BENCHMARK LOW HIGH | 3.5 1.8 7.0 | 13.51 13.51 13.33 | 2 2 2 | 27.03 27.03 26.67 | 100.0% 100.0% 98.7% |
| TIME CONSTANT OF DENDRITIC POTASSIUM CURRENT (3) | TGKD | BENCHMARK LOW HIGH | 10.0 5.0 20.0 | 13.51 21.74 8.00 | 2 2 3 | 27.03 43.48 24.00 | 100.0% 160.9% 88.8% |
| ACTIVATION RATE OF CALCIUM CONDUCTANCE | D | BENCHMARK LOW HIGH | 2.2 1.1 4.4 | 13.51 14.71 11.11 | 2 2 4 | 27.03 29.41 44.44 | 100.0% 108.8% 164.4% |
| TIME CONSTANT OF DENDRITIC CALCIUM CONDUCTANCE | TGC | BENCHMARK LOW HIGH | 5.0 2.5 10.0 | 13.51 14.29 12.82 | 2 2 2 | 27.03 28.57 25.64 | 100.0% 105.7% 94.9% |
| ACCUMULATION RATE OF CALCIUM FOR A GIVEN CALCIUM CONDUCTANCE (4) | A | BENCHMARK LOW HIGH | 2.0 1.0 4.0 | 13.51 13.51 12.99 | 2 3 1 | 27.03 40.54 12.99 | 100.0% 150.0% 48.1% |
| TIME CONSTANT FOR CALCIUM ACCUMULATION | TCA | BENCHMARK LOW HIGH | 5.0 2.5 10.0 | 13.51 14.71 11.76 | 2 1 3 | 27.03 14.71 35.29 | 100.0% 54.4% 130.6% |
| INPUT CONDUCTANCE FROM DENDRITE TO SOMA (5) | GDS | BENCHMARK LOW HIGH | 5.0 2.5 10.0 | 13.51 11.90 14.29 | 2 1 4 | 27.03 11.90 57.14 | 100.0% 44.0% 211.4% |
| INPUT CONDUCTANCE FROM SOMA TO DENDRITE | GSD | BENCHMARK LOW HIGH | 5.0 2.5 10.0 | 13.51 13.89 10.75 | 2 2 2 | 27.03 27.78 21.51 | 100.0% 102.8% 79.6% |
| SOMATIC FIRING THRESHOLD | THRESHOLD | BENCHMARK LOW HIGH | 12.0 6.0 24.0 | 13.51 15.38 13.16 | 2 4 1 | 27.03 61.54 13.16 | 100.0% 227.7% 48.7% |
| CA SPIKE THRESHOLD IN DENDRITE (6) | CSPKTHRESH | BENCHMARK LOW HIGH | 12.0 6.0 24.0 | 13.51 14.08 13.70 | 2 2 2 | 27.03 28.17 27.40 | 100.0% 104.2% 101.4% |
| SYNAPTIC INPUT TO DENDRITE (8) | INPUT | BENCHMARK LOW (7) HIGH | 35.0 27.0 70.0 | 13.51 11.63 16.95 | 2 2 2 | 27.03 23.26 33.90 | 100.0% 86.0% 125.4% |

## NOTES TO PARAMETER SENSITIVITY ANALYSIS

(1) The number of spikes per burst is altered by modulating the internal calcium concentration required to trigger the dendritic potassium current. In an observation repeated several times herein, it seems clear that modulating the hyperpolarizing potassium current has a marked effectiveness in modulating the neuron's output.

(2) Modulating the activation rate (B) of the somatic potassium current strongly effects firing, but changing the time constant of this current has almost no effect either on bursts/second or spikes/burst.

(3) However, note that, among all 17 parameters of this model neuron, it is only the time constant of the [CA]-gated dendritic potassium current which is effective in modulating the rate of bursting (whereas the somatic potassium current time constant does not seem to effect the model neuron's output at all).

(4) This quantity, the accumulation rate of calcium in the dendrite per unit calcium conductance, would increase as the effectiveness of calcium buffers within the dendrite decreased.

(5) Despite its efficacy in modulating the neuron's output, this parameter is presumably not a likely candidate for plasticity, because it depends on the axial resistance of the cytoplasm, the cross section of the base of the dendrite, and the volume of the soma, all of which seem unlikely to be the subject to modulation.

(6) Surprisingly, the overall input-output relation for the neuron is not much effected by changing the threshold for the voltage gated calcium spike activity in the dendrite.

(7) The minimum dendritic input required to produce any spike activity (that is, to increase the voltage in the soma above firing threshold) may be calculated to be 26.4 with all the other parameters at benchmark values. Hence 27 is an input level that is only 2% above the minimum level to get any firing at all. Note that it appears a 2 spike burst is always produced (with the internal parameters set at the benchmark levels) if any firing at all is elicited. The number of spikes per burst, then, is modulated by conductance activation rates and calcium accumulation rates but not by input. Tables 2 and 3 demonstrate this over a wide range of inputs.

(8) Note that doubling the synaptic input to the dendrite only increases the model neuron's firing rate by 25.4%, but that, for example, doubling the activiation rate of the dendritic calcium current increases the firing rate by 64.4%. Hence we suggest that modulation of synaptic efficacy is not the only choice or even the most effective choice for the mechanism underlying plasticity. Alkon (1988,1989) and others have in fact recently reported that an increase in protein kinase C, leading to a reduction in calcium-activated potassium current, is observed to be associated with conditioning in Hermissenda and rabbit. Thus, plasticity in the nervous system may indeed operate via a whole set of internal dynamic parameters, of which synapse efficacy is only one.

# DENDRITIC K–CURRENT TIME CONST: 5 MSEC
## FIRING RATE=43.48    BURST RATE=21.74

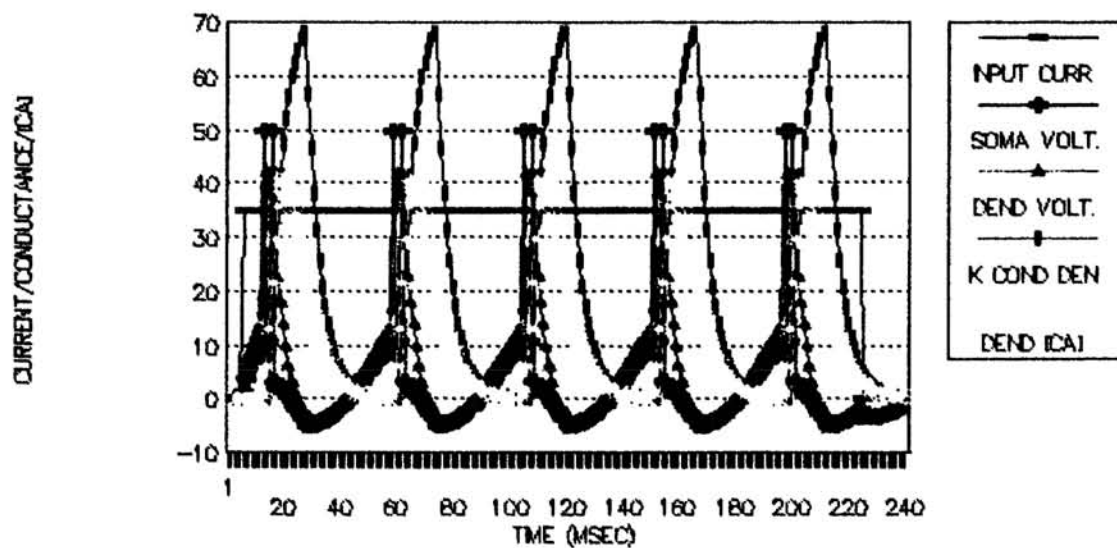

# DENDRITIC K–CURRENT TIME CONST: 20 MSEC
## FIRING RATE=24.00    BURST RATE=8.00

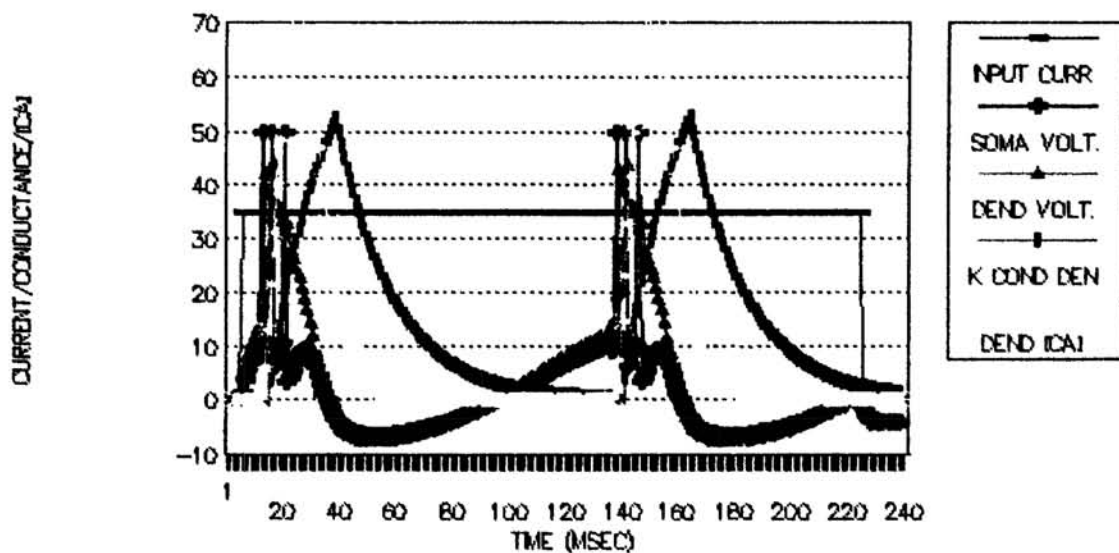

Figure 1

# THE INPUT/OUTPUT RELATION
# CA ACCUMULATION RATE SET AT 3 LEVELS

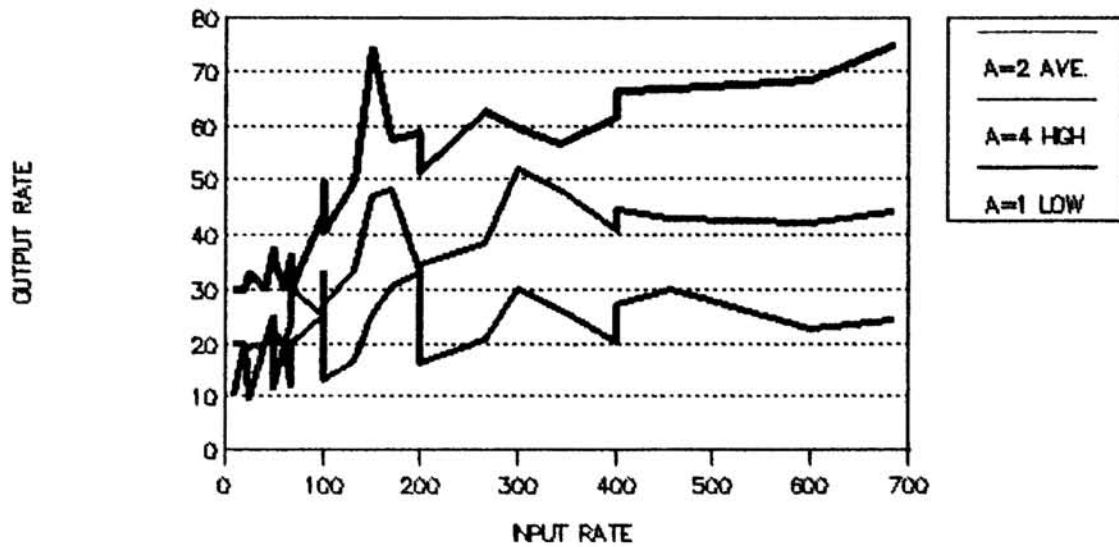

Figure 2

# COMPARISON OF INPUT/OUTPUT RELATION
# EPSP SQUARE PULSE VS ALPHA FUNCTION

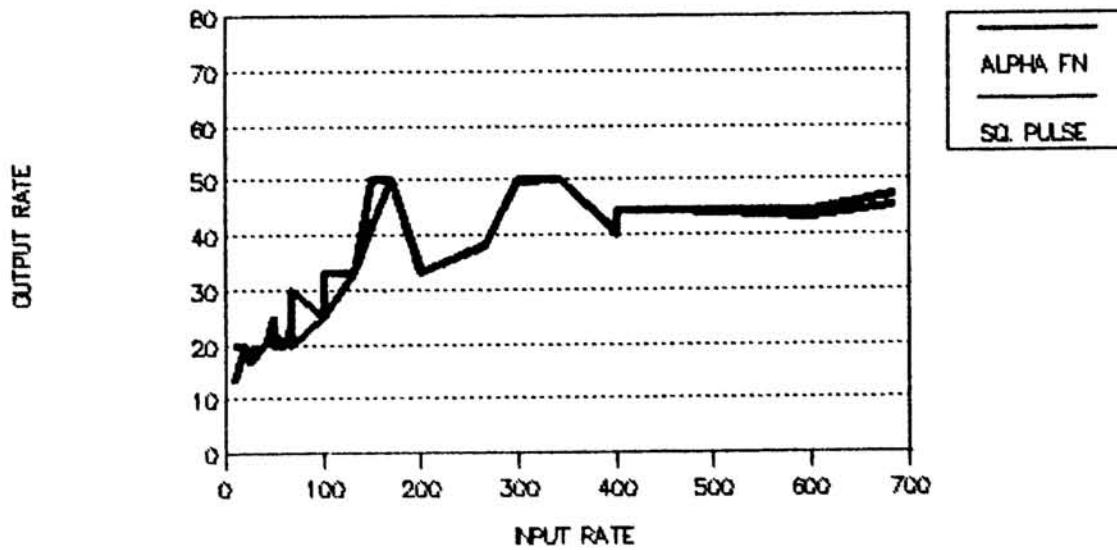

Figure 3